# Syntactic Topic Models

**Jordan Boyd-Graber**
Department of Computer Science
35 Olden Street
Princeton University
Princeton, NJ 08540
jbg@cs.princeton.edu

**David Blei**
Department of Computer Science
35 Olden Street
Princeton University
Princeton, NJ 08540
blei@cs.princeton.edu

## Abstract

We develop the syntactic topic model (STM), a nonparametric Bayesian model of parsed documents. The STM generates words that are both thematically and syntactically constrained, which combines the semantic insights of topic models with the syntactic information available from parse trees. Each word of a sentence is generated by a distribution that combines document-specific topic weights and parse-tree-specific syntactic transitions. Words are assumed to be generated in an order that respects the parse tree. We derive an approximate posterior inference method based on variational methods for hierarchical Dirichlet processes, and we report qualitative and quantitative results on both synthetic data and hand-parsed documents.

## 1   Introduction

Probabilistic topic models provide a suite of algorithms for finding low dimensional structure in a corpus of documents. When fit to a corpus, the underlying representation often corresponds to the "topics" or "themes" that run through it. Topic models have improved information retrieval [1], word sense disambiguation [2], and have additionally been applied to non-text data, such as for computer vision and collaborative filtering [3, 4].

Topic models are widely applied to text despite a willful ignorance of the underlying linguistic structures that exist in natural language. In a topic model, the words of each document are assumed to be *exchangeable*; their probability is invariant to permutation. This simplification has proved useful for deriving efficient inference techniques and quickly analyzing very large corpora [5].

However, exchangeable word models are limited. While useful for classification or information retrieval, where a coarse statistical footprint of the themes of a document is sufficient for success, exchangeable word models are ill-equipped for problems relying on more fine-grained qualities of language. For instance, although a topic model can suggest documents relevant to a query, it cannot find particularly relevant phrases for question answering. Similarly, while a topic model might discover a pattern such as "eat" occurring with "cheesecake," it lacks the representation to describe selectional preferences, the process where certain words restrict the choice of the words that follow.

It is in this spirit that we develop the *syntactic topic model*, a nonparametric Bayesian topic model that can infer both syntactically and thematically coherent topics. Rather than treating words as the exchangeable unit within a document, the words of the sentences must conform to the structure of a parse tree. In the generative process, the words arise from a distribution that has both a document-specific thematic component and a parse-tree-specific syntactic component.

We illustrate this idea with a concrete example. Consider a travel brochure with the sentence "In the near future, you could find yourself in ____." Both the low-level syntactic context of a word and its document context constrain the possibilities of the word that can appear next. Syntactically, it

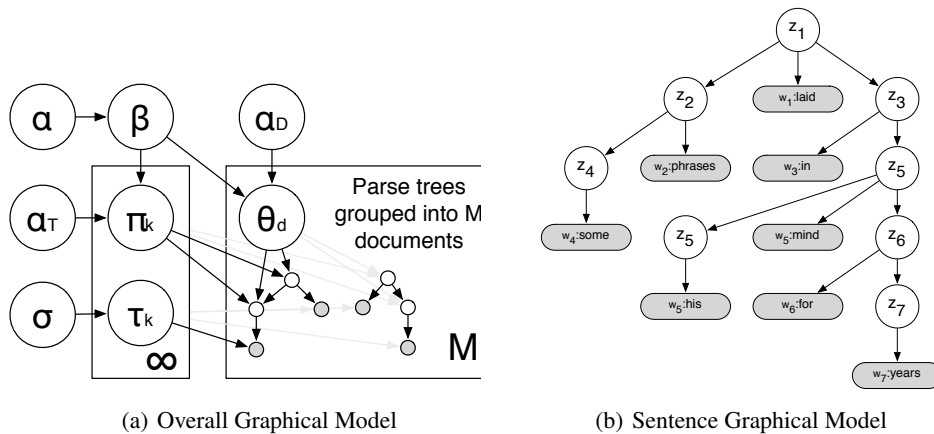

(a) Overall Graphical Model        (b) Sentence Graphical Model

Figure 1: In the graphical model of the STM, a document is made up of a number of sentences, represented by a tree of latent topics $z$ which in turn generate words $w$. These words' topics are chosen by the topic of their parent (as encoded by the tree), the topic weights for a document $\theta$, and the node's parent's successor weights $\pi$. (For clarity, not all dependencies of sentence nodes are shown.) The structure of variables for sentences within the document plate is on the right, as demonstrated by an automatic parse of the sentence "Some phrases laid in his mind for years." The STM assumes that the tree structure and words are given, but the latent topics $z$ are not.

is going to be a noun consistent as the object of the preposition "of." Thematically, because it is in a travel brochure, we would expect to see words such as "Acapulco," "Costa Rica," or "Australia" more than "kitchen," "debt," or "pocket." Our model can capture these kinds of regularities and exploit them in predictive problems.

Previous efforts to capture local syntactic context include semantic space models [6] and similarity functions derived from dependency parses [7]. These methods successfully determine words that share similar contexts, but do not account for thematic consistency. They have difficulty with polysemous words such as "fly," which can be either an insect or a term from baseball. With a sense of document context, i.e., a representation of whether a document is about sports or animals, the meaning of such terms can be distinguished.

Other techniques have attempted to combine local context with document coherence using linear sequence models [8, 9]. While these models are powerful, ordering words sequentially removes the important connections that are preserved in a syntactic parse. Moreover, these models generate words either from the syntactic or thematic context. In the syntactic topic model, words are constrained to be consistent with both.

The remainder of this paper is organized as follows. We describe the syntactic topic model, and develop an approximate posterior inference technique based on variational methods. We study its performance both on synthetic data and hand parsed data [10]. We show that the STM captures relationships missed by other models and achieves lower held-out perplexity.

## 2 The syntactic topic model

We describe the syntactic topic model (STM), a document model that combines observed syntactic structure and latent thematic structure. To motivate this model, we return to the travel brochure sentence "In the near future, you could find yourself in ____.". The word that fills in the blank is constrained by its syntactic context and its document context. The syntactic context tells us that it is an object of a preposition, and the document context tells us that it is a travel-related word.

The STM attempts to capture these joint influences on words. It models a document corpus as exchangeable collections of sentences, each of which is associated with a tree structure such as a

parse tree (Figure 1(b)). The words of each sentence are assumed to be generated from a distribution influenced both by their observed role in that tree and by the latent topics inherent in the document.

The latent variables that comprise the model are topics, topic transition vectors, topic weights, topic assignments, and top-level weights. *Topics* are distributions over a fixed vocabulary ($\tau_k$ in Figure 1). Each is further associated with a *topic transition vector* ($\pi_k$), which weights changes in topics between parent and child nodes. *Topic weights* ($\theta_d$) are per-document vectors indicating the degree to which each document is "about" each topic. *Topic assignments* ($z_n$, associated with each internal node of 1(b)) are per-word indicator variables that refer to the topic from which the corresponding word is assumed to be drawn. The STM is a nonparametric Bayesian model. The number of topics is not fixed, and indeed can grow with the observed data.

The STM assumes the following generative process of a document collection.

1. Choose global topic weights $\beta \sim \text{GEM}(\alpha)$
2. For each topic index $k = \{1, \dots\}$:
   (a) Choose topic $\tau_k \sim \text{Dir}(\sigma)$
   (b) Choose topic transition distribution $\pi_k \sim \text{DP}(\alpha_T, \beta)$
3. For each document $d = \{1, \dots M\}$:
   (a) Choose topic weights $\theta_d \sim \text{DP}(\alpha_D, \beta)$
   (b) For each sentence in the document:
       i. Choose topic assignment $z_0 \propto \theta_d \pi_{start}$
       ii. Choose root word $w_0 \sim \text{mult}(1, \tau_{z_0})$
       iii. For each additional word $w_n$ and parent $p_n$, $n \in \{1, \dots d_n\}$
           • Choose topic assignment $z_n \propto \theta_d \pi_{z_{p(n)}}$
           • Choose word $w_n \sim \text{mult}(1, \tau_{z_n})$

The distinguishing feature of the STM is that the topic assignment is drawn from a distribution that combines two vectors: the per-document topic weights and the transition probabilities of the topic assignment from its parent node in the parse tree. By merging these vectors, the STM models both the local syntactic context and corpus-level semantics of the words in the documents. Because they depend on their parents, the topic assignments and words are generated by traversing the tree.

A natural alternative model would be to traverse the tree and choose the topic assignment from either the parental topic transition $\pi_{z_{p(n)}}$ or document topic weights $\theta_d$, based on a binary selector variable. This would be an extension of [8] to parse trees, but it does not enforce words to be syntactically consistent with their parent nodes *and* be thematically consistent with a topic of the document. Only one of the two conditions must be true. Rather, this approach draws on the idea behind the product of experts [11], multiplying two vectors and renormalizing to obtain a new distribution. Taking the point-wise product can be thought of as viewing one distribution through the "lens" of another, effectively choosing only words whose appearance can be explained by both.

The STM is closely related to the hierarchical Dirichlet process (HDP). The HDP is an extension of Dirichlet process mixtures to grouped data [12]. Applied to text, the HDP is a probabilistic topic model that allows each document to exhibit multiple topics. It can be thought of as the "infinite" topic version of latent Dirichlet allocation (LDA) [13]. The difference between the STM and the HDP is in how the per-word topic assignment is drawn. In the HDP, this topic assignment is drawn directly from the topic weights and, thus, the HDP assumes that words within a document are exchangeable. In the STM, the words are generated conditioned on their parents in the parse tree. The exchangeable unit is a sentence.

The STM is also closely related to the infinite tree with independent children [14]. The infinite tree models syntax by basing the latent syntactic category of children on the syntactic category of the parent. The STM reduces to the Infinite Tree when $\theta_d$ is fixed to a vector of ones.

## 3   Approximate posterior inference

The central computational problem in topic modeling is to compute the posterior distribution of the latent structure conditioned on an observed collection of documents. Specifically, our goal is to compute the posterior topics, topic transitions, per-document topic weights, per-word topic assign-

ments, and top-level weights conditioned on a set of documents, each of which is a collection of parse trees.

This posterior distribution is intractable to compute. In typical topic modeling applications, it is approximated with either variational inference or collapsed Gibbs sampling. Fast Gibbs sampling relies on the conjugacy between the topic assignment and the prior over the distribution that generates it. The syntactic topic model does not enjoy such conjugacy because the topic assignment is drawn from a multiplicative combination of two Dirichlet distributed vectors. We appeal to variational inference.

In variational inference, the posterior is approximated by positing a simpler family of distributions, indexed by free variational parameters. The variational parameters are fit to be close in relative entropy to the true posterior. This is equivalent to maximizing the Jensen's bound on the marginal probability of the observed data [15].

We use a fully-factorized variational distribution,

$$q(\beta, z, \theta, \pi, \tau | \beta^*, \phi, \gamma, \nu) = q(\beta|\beta^*) \prod_d q(\theta_d|\gamma_d) \prod_k q(\pi_k|\nu_k) \prod_n q(z_n|\phi_n). \quad (1)$$

Following [16], $q(\beta|\beta^*)$ is not a full distribution, but is a degenerate point estimate truncated so that all weights whose index is greater than $K$ are zero in the variational distribution. The variational parameters $\gamma_d$ and $\nu_z$ index Dirichlet distributions, and $\phi_n$ is a topic multinomial for the $n^{th}$ word.

From this distribution, the Jensen's lower bound on the log probability of the corpus is $L(\gamma, \nu, \phi; \beta, \theta, \pi, \tau) =$

$$\mathbb{E}_q \left[ \log p(\beta|\alpha) + \log p(\boldsymbol{\theta}|\alpha_D, \beta) + \log p(\boldsymbol{\pi}|\alpha_P, \beta) + \log p(\boldsymbol{z}|\boldsymbol{\theta}, \boldsymbol{\pi}) + \right.$$
$$\left. \log p(\boldsymbol{w}|\boldsymbol{z}, \boldsymbol{\tau}) + \log p(\boldsymbol{\tau}|\sigma) \right] - \mathbb{E}_q \left[ \log q(\boldsymbol{\theta}) + \log q(\boldsymbol{\pi}) + \log q(\boldsymbol{z}) \right]. \quad (2)$$

Expanding $\mathbb{E}_q \left[ \log p(\mathbf{z}|\theta, \pi) \right]$ is difficult, so we add an additional slack parameter, $\omega_n$ to approximate the expression. This derivation and the complete likelihood bound is given in the supplement. We use coordinate ascent to optimize the variational parameters to be close to the true posterior.

**Per-word variational updates** The variational update for the topic assignment of the $n$th word is

$$
\begin{aligned}
\phi_{ni} \quad \propto \quad & \exp \left\{ \Psi(\gamma_i) - \Psi(\textstyle\sum_{j=1}^K \gamma_j) + \sum_{j=1}^K \phi_{p(n),j} \left( \Psi(\nu_{j,i}) - \Psi \left( \sum_{k=1}^K \nu_{j,k} \right) \right) \right. \\
& + \sum_{c \in c(n)} \sum_{j=1}^K \phi_{c,j} \left( \Psi(\nu_{i,j}) - \Psi \left( \sum_{k=1}^K \nu_{i,k} \right) \right) \\
& \left. - \sum_{c \in c(n)} \omega_c^{-1} \sum_j^K \frac{\gamma_j \nu_{i,j}}{\sum_k \gamma_k \sum_k \nu_{i,k}} + \log \tau_{i,w_n} \right\}. \quad (3)
\end{aligned}
$$

The influences on estimating the posterior of a topic assignment are: the document's topic $\gamma$, the topic of the node's parent $p(n)$, the topic of the node's children $c(n)$, the expected transitions between topics $\nu$, and the probability of the word within a topic $\tau_{i,w_n}$.

Most terms in Equation 3 are familiar from variational inference for probabilistic topic models, as the digamma functions appear in the expectations of multinomial distributions. The second to last term is new, however, because we cannot assume that the point-wise product of $\pi_k$ and $\theta_d$ will sum to one. We approximate the normalizer for their produce by introducing $\omega$; its update is

$$\omega_n = \sum_{i=1}^K \sum_{j=1}^K \phi_{p(n),j} \frac{\gamma_i \nu_{j,i}}{\sum_{k=1}^K \gamma_k \sum_{k=1}^K \nu_{j,k}}.$$

**Variational Dirichlet distributions and topic composition** This normalizer term also appears in the derivative of the likelihood function for $\gamma$ and $\nu$ (the parameters to the variational distributions on $\theta$ and $\pi$, respectively), which cannot be solved in a closed form. We use conjugate gradient optimization to determine the appropriate updates for these parameters [17].

**Top-level weights** Finally, we consider the top-level weights. The first $K - 1$ stick-breaking proportions are drawn from a Beta distribution with parameters $(1, \alpha)$, but we assume that the final stick-breaking proportion is unity (thus implying $\beta^*$ is non-zero only from $1 \ldots K$). Thus, we only optimize the first $K - 1$ positions and implicitly take $\beta_K^* = 1 - \sum_i^{K-1} \beta_i^*$. This constrained optimization is performed using the barrier method [17].

# 4 Empirical results

Before considering real-world data, we demonstrate the STM on synthetic natural language data. We generated synthetic sentences composed of verbs, nouns, prepositions, adjectives, and determiners. Verbs were only in the head position; prepositions could appear below nouns or verbs; nouns only appeared below verbs; prepositions or determiners and adjectives could appear below nouns. Each of the parts of speech except for prepositions and determiners were sub-grouped into themes, and a document contains a single theme for each part of speech. For example, a document can only contain nouns from a single "economic," "academic," or "livestock" theme.

Using a truncation level of 16, we fit three different nonparametric Bayesian language models to the synthetic data (Figure 2).[1] The infinite tree model is aware of the tree structure but not documents [14] It is able to separate parts of speech successfully except for adjectives and determiners (Figure 2(a)). However, it ignored the thematic distinctions that actually divided the terms between documents. The HDP is aware of document groupings and treats the words exchangeably within them [12]. It is able to recover the thematic topics, but has missed the connections between the parts of speech, and has conflated multiple parts of speech (Figure 2(b)).

The STM is able to capture the the topical themes and recover parts of speech (with the exception of prepositions that were placed in the same topic as nouns with a self loop). Moreover, it was able to identify the same interconnections between latent classes that were apparent from the infinite tree. Nouns are dominated by verbs and prepositions, and verbs are the root (head) of sentences.

**Qualitative description of topics learned from hand-annotated data**    The same general properties, but with greater variation, are exhibited in real data. We converted the Penn Treebank [10], a corpus of manually curated parse trees, into a dependency parse [18]. The vocabulary was pruned to terms that appeared in at least ten documents.

Figure 3 shows a subset of topics learned by the STM with truncation level 32. Many of the resulting topics illustrate both syntactic and thematic consistency. A few nonspecific function topics emerged (pronoun, possessive pronoun, general verbs, etc.). Many of the noun categories were more specialized. For instance, Figure 3 shows clusters of nouns relating to media, individuals associated with companies ("mr," "president," "chairman"), and abstract nouns related to stock prices ("shares," "quarter," "earnings," "interest"), all of which feed into a topic that modifies nouns ("his," "their," "other," "last"). Thematically related topics are separated by both function and theme.

This division between functional and topical uses for the latent classes can also been seen in the values for the per-document multinomial over topics. A number of topics in Figure 3(b), such as 17, 15, 10, and 3, appear to some degree in nearly every document, while other topics are used more sparingly to denote specialized content. With $\alpha = 0.1$, this plot also shows that the nonparametric Bayesian framework is ignoring many later topics.

**Perplexity**    To study the performance of the STM on new data, we estimated the held out probability of previously unseen documents with an STM trained on a portion of the Penn Treebank. For each position in the parse trees, we estimate the probability the observed word. We compute the perplexity as the exponent of the inverse of the per-word average log probability. The lower the perplexity, the better the model has captured the patterns in the data. We also computed perplexity for individual parts of speech to study the differences in predictive power between content words, such as nouns and verbs, and function words, such as prepositions and determiners. This illustrates how different algorithms better capture aspects of context. We expect function words to be dominated by local context and content words to be determined more by the themes of the document.

This trend is seen not only in the synthetic data (Figure 4(a)), where parsing models better predict functional categories like prepositions and document only models fail to account for patterns of verbs and determiners, but also in real data. Figure 4(b) shows that HDP and STM both perform better than parsing models in capturing the patterns behind nouns, while both the STM and the infinite tree have lower perplexity for verbs. Like parsing models, our model was better able to

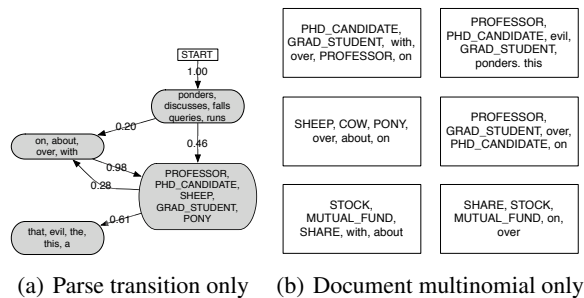

(a) Parse transition only     (b) Document multinomial only

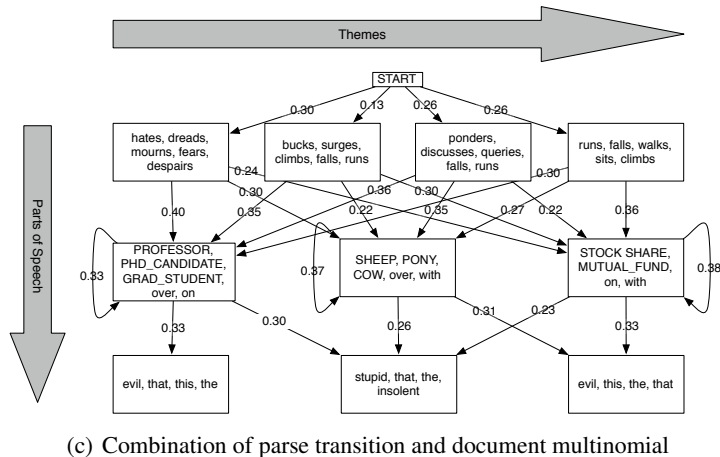

(c) Combination of parse transition and document multinomial

Figure 2: Three models were fit to the synthetic data described in Section 4. Each box illustrates the top five words of a topic; boxes that represent homogenous parts of speech have rounded edges and are shaded. Edges between topics are labeled with estimates of their transition weight $\pi$. While the infinite tree model (a) is able to reconstruct the parts of speech used to generate the data, it lumps all topics into the same categories. Although the HDP (b) can discover themes of recurring words, it cannot determine the interactions between topics or separate out ubiquitous words that occur in all documents. The STM (c) is able to recover the structure.

predict the appearance of prepositions, but also remained competitive with HDP on content words. On the whole, the STM had lower perplexity than HDP and the infinite tree.

## 5 Discussion

We have introduced and evaluated the syntactic topic model, a nonparametric Bayesian model of parsed documents. The STM achieves better perplexity than the infinite tree or the hierarchical Dirichlet process and uncovers patterns in text that are both syntactically and thematically consistent. This dual relevance is useful for work in natural language processing. For example, recent work [19, 20] in the domain of word sense disambiguation has attempted to combine syntactic similarity with topical information in an ad hoc manner to improve the predominant sense algorithm [21]. The syntactic topic model offers a principled way to learn both simultaneously rather than combining two heterogenous methods.

The STM is not a full parsing model, but it could be used as a means of integrating document context into parsing models. This work's central premise is consistent with the direction of recent improvements in parsing technology in that it provides a method for refining the parts of speech present in a corpus. For example, lexicalized parsers [22] create rules specific to individual terms, and grammar refinement [23] divides general roles into multiple, specialized ones. The syntactic topic model offers an alternative method of finding more specific rules by grouping words together that appear in similar documents and could be extended to a full parser.

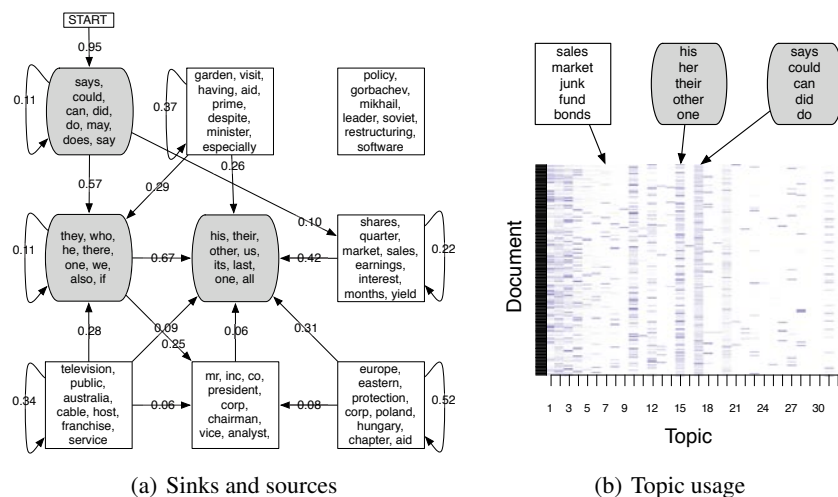

(a) Sinks and sources                                    (b) Topic usage

Figure 3: Selected topics (along with strong links) after a run of the syntactic topic model with a truncation level of 32. As in Figure 2, parts of speech that aren't subdivided across themes are indicated. In the Treebank corpus (left), head words (verbs) are shared, but the nouns split off into many separate specialized categories before feeding into pronoun sinks. The specialization of topics is also visible in plots of the variational parameter $\gamma$ normalized for the first 300 documents of the Treebank (right), where three topics columns have been identified. Many topics are used to some extent in every document, showing that they are performing a functional role, while others are used more sparingly for semantic content.

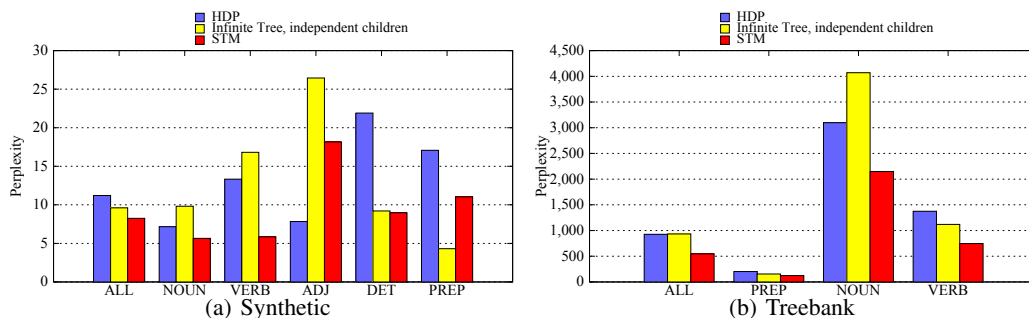

(a) Synthetic                                    (b) Treebank

Figure 4: After fitting three models on synthetic data, the syntactic topic model has better (lower) perplexity on all word classes except for adjectives. HDP is better able to capture document-level patterns of adjectives. The infinite tree captures prepositions best, which have no cross-document variation. On real data 4(b), the syntactic topic model was able to combine the strengths of the infinite tree on functional categories like prepositions with the strengths of the HDP on content categories like nouns to attain lower overall perplexity.

While traditional topic models reveal groups of words that are used in similar documents, the STM uncovers groups that are used the same way in similar documents. This decomposition is useful for tasks that require a more fine-grained representation of language than the bag of words can offer or for tasks that require a broader context than parsing models.

## Footnotes

[1]In Figure 2 and Figure 3, we mark topics which represent a single part of speech and are essentially the lone representative of that part of speech in the model. This is a subjective determination of the authors, does not reflect any specialization or special treatment of topics by the model, and is done merely for didactic purposes.

# References

[1] Wei, X., B. Croft. LDA-based document models for ad-hoc retrieval. In *Proceedings of the ACM SIGIR Conference on Research and Development in Information Retrieval*. 2006.

[2] Cai, J. F., W. S. Lee, Y. W. Teh. NUS-ML:Improving word sense disambiguation using topic features. In *Proceedings of SemEval-2007*. Association for Computational Linguistics, 2007.

[3] Fei-Fei Li, P. Perona. A Bayesian hierarchical model for learning natural scene categories. In *CVPR '05 - Volume 2*, pages 524–531. IEEE Computer Society, Washington, DC, USA, 2005.

[4] Marlin, B. Modeling user rating profiles for collaborative filtering. In S. Thrun, L. Saul, B. Schölkopf, eds., *Advances in Neural Information Processing Systems*. MIT Press, Cambridge, MA, 2004.

[5] Griffiths, T., M. Steyvers. Probabilistic topic models. In T. Landauer, D. McNamara, S. Dennis, W. Kintsch, eds., *Latent Semantic Analysis: A Road to Meaning*. Laurence Erlbaum, 2006.

[6] Padó, S., M. Lapata. Dependency-based construction of semantic space models. *Computational Linguistics*, 33(2):161–199, 2007.

[7] Lin, D. An information-theoretic definition of similarity. In *Proceedings of International Conference of Machine Learning*, pages 296–304. 1998.

[8] Griffiths, T. L., M. Steyvers, D. M. Blei, et al. Integrating topics and syntax. In L. K. Saul, Y. Weiss, L. Bottou, eds., *Advances in Neural Information Processing Systems*, pages 537–544. MIT Press, Cambridge, MA, 2005.

[9] Gruber, A., M. Rosen-Zvi, Y. Weiss. Hidden topic Markov models. In *Proceedings of Artificial Intelligence and Statistics*. San Juan, Puerto Rico, 2007.

[10] Marcus, M. P., B. Santorini, M. A. Marcinkiewicz. Building a large annotated corpus of English: The Penn treebank. *Computational Linguistics*, 19(2):313–330, 1994.

[11] Hinton, G. Products of experts. In *Proceedings of the Ninth International Conference on Artificial Neural Networks*, pages 1–6. IEEE, Edinburgh, Scotland, 1999.

[12] Tee, Y. W., M. I. Jordan, M. J. Beal, et al. Hierarchical dirichlet processes. *Journal of the American Statistical Association*, 101(476):1566–1581, 2006.

[13] Blei, D., A. Ng, M. Jordan. Latent Dirichlet allocation. *Journal of Machine Learning Research*, 3:993–1022, 2003.

[14] Finkel, J. R., T. Grenager, C. D. Manning. The infinite tree. In *Proceedings of Association for Computational Linguistics*, pages 272–279. Association for Computational Linguistics, Prague, Czech Republic, 2007.

[15] Jordan, M., Z. Ghahramani, T. S. Jaakkola, et al. An introduction to variational methods for graphical models. *Machine Learning*, 37(2):183–233, 1999.

[16] Liang, P., S. Petrov, M. Jordan, et al. The infinite PCFG using hierarchical Dirichlet processes. In *Proceedings of Emperical Methods in Natural Language Processing*, pages 688–697. 2007.

[17] Boyd, S., L. Vandenberghe. *Convex Optimization*. Cambridge University Press, 2004.

[18] Johansson, R., P. Nugues. Extended constituent-to-dependency conversion for English. In *(NODALIDA)*. 2007.

[19] Koeling, R., D. McCarthy. Sussx: WSD using automatically acquired predominant senses. In *Proceedings of SemEval-2007*. Association for Computational Linguistics, 2007.

[20] Boyd-Graber, J., D. Blei. PUTOP: Turning predominant senses into a topic model for WSD. In *Proceedings of SemEval-2007*. Association for Computational Linguistics, 2007.

[21] McCarthy, D., R. Koeling, J. Weeds, et al. Finding predominant word senses in untagged text. In *Proceedings of Association for Computational Linguistics*, pages 280–287. Association for Computational Linguistics, 2004.

[22] Collins, M. Head-driven statistical models for natural language parsing. *Computational Linguistics*, 29(4):589–637, 2003.

[23] Klein, D., C. Manning. Accurate unlexicalized parsing. In *Proceedings of Association for Computational Linguistics*, pages 423–430. Association for Computational Linguistics, 2003.

